# Dynamic Modelling of Chaotic Time Series with Neural Networks

Jose C. Principe, Jyh-Ming Kuo
Computational NeuroEngineering Laboratory
University of Florida, Gainesville, FL32611
principe@synapse.ee.ufl.edu

## Abstract

This paper discusses the use of artificial neural networks for dynamic modelling of time series. We argue that multistep prediction is more appropriate to capture the dynamics of the underlying dynamical system, because it constrains the iterated model. We show how this method can be implemented by a recurrent ANN trained with trajectory learning. We also show how to select the trajectory length to train the iterated predictor for the case of chaotic time series. Experimental results corroborate the proposed method.

## 1.0 Introduction

The search for a model of an experimental time series has been an important problem in science. For a long time the linear model was almost exclusively used to describe the system that produced the time series [1], but recently nonlinear models have also been proposed to replace the linear ones [2]. Lapedes and Farber [3] showed how artificial neural networks (ANNs) can be used to identify the dynamics of the unknown system that produced the time series. He simply used a multilayer perceptron to predict the next point in state space, and trained this topology with backpropagation. This paper explores more complex neural topologies and training methods with the goal of improving the quality of the identification of the dynamical system, and to understand better the issues of dynamic modelling with neural networks which are far from being totally understood.

According to Takens' embedding theorem, a map $F: R^{2m+1} \to R^{2m+1}$ exists that transforms the current reconstructed state $\overset{\rightarrow}{y}(t)$ to the next state $\overset{\rightarrow}{y}(t+1)$, i.e.

$$\overset{\rightarrow}{y}(t+1) = F(\overset{\rightarrow}{y}(t)) \qquad (1)$$

or

$$\begin{bmatrix} x(t+1) \\ \dots \\ x(t+1-2m) \end{bmatrix} = F\left( \begin{bmatrix} x(t) \\ \dots \\ x(t-2m) \end{bmatrix} \right)$$

where m is the estimated dimension of the unknown dynamical system $\Phi$. Note that the map contains several trivial (nonlinear) filters and a predictor. The predictive mapping $F^{\perp}:R^{2m+1} \rightarrow R$ can be expressed as

$$x(t+1) = F^{\perp}(\dot{x}(t)) \tag{2}$$

where $\dot{x}(t) = [x(t-2m)\dots x(t-1)x(t)]^{T}$. This is actually the estimated nonlinear autoregressive model of the input time series. *The existence of this predictive model lays a theoretical basis for dynamic modelling in the sense that we can build from a vector time series a model to approximate the mapping $F^{\perp}$.* If the conditions of Takens embedding theorem are met, this mapping captures some of the properties of the unknown dynamical system $\Phi$ that produced the time series [7].

Presently one still does not have a capable theory to guarantee if the predictor has successfully identified the original model $\Phi$. The simple point by point comparison between the original and predicted time series used as goodness of fit for non-chaotic time series breaks down for chaotic ones [5]. Two chaotic time series can be very different pointwise but be produced by the same dynamical system (two trajectories around the same attractor). The dynamic invariants (correlation dimension, Lyapunov exponents) measure global properties of the attractor, so they should be used as the rule to decide about the success of dynamic modelling. Hence, *a pragmatic approach in dynamic modelling is to seed the predictor with a point in state space, feed the output to its input as an autonomous system, and create a new time series.* If the dynamic invariants computed from this time series match the ones from the original time series, then we say that dynamic modelling was successful [5]. The long term behavior of the autonomous predictive model seems to be the key factor to find out if the predictor identified the original model. This is the distinguishing factor between prediction of chaotic time series and dynamic modelling. The former only addresses the instantaneous prediction error, while the latter is interested in long term behavior.

In order to use this theory, one needs to address the choices of predictor implementation. Due to the universal mapping characteristics of multilayer perceptrons (MLPs) and the existence of well established learning rules to adapt the MLP coefficients, this type of network appears as an appropriate choice [3]. However, one must realize that the MLP is a static mapper, and in dynamic modelling we are dealing with time varying signals, where the past of the signal contains vital information to describe the mapping. The design considerations to select the neural network topology are presented elsewhere [4]. We just would like to say that the MLP has to be enhanced with short term memory mechanisms, and that the *estimation of the correlation dimension should be used to set the size of the memory layer.* The main goal of the paper is to establish the methodology to efficiently train neural networks for dynamic modelling.

## 2. Iterated versus Single Step Prediction.

From eqn. 2 it seems that the resulting dynamic model F can be obtained through single step prediction. This has been the conventional way to handle dynamic modelling [2],[3]. The predictor is adapted by minimizing the error

$$E = \sum_{i=2m+1}^{L} dist(x(i+1) - \tilde{F}^{\perp}(\dot{x}(i)))$$ (3)

where L is the length of the time series, x(i) is the i[th] data sample, $\tilde{F}^{\perp}$ is the map developed by the predictor and *dist()* is a distance measure (normally the L2 norm). Notice that the training to obtain the mapping is done independently from sample to sample, i.e.

$$x(i+1) = \tilde{F}^{\perp}(\dot{x}(i)) + \delta_1$$

$$\cdots$$

$$x(i+j) = \tilde{F}^{\perp}(\dot{x}(i+j-1)) + \delta_j$$

where $\delta_j$ are the instantaneous prediction errors, which are minimized during training. Notice that the predictor is being optimized under the assumption that the previous point in state space is known without error.

The problem with this approach can be observed when we iterate the predictor as an autonomous system to generate the time series samples. If one wants to produce two samples in the future from sample i the predicted sample i+1 needs to be utilized to generate sample i+2. The predictor was not optimized to do this job, because during training the true i+1 sample was assumed known. As long as $\delta_1$ is nonzero (as will be always the case for nontrivial problems), errors will accumulate rapidly. Single step prediction is more associated with extrapolation than with dynamic modelling, which requires the identification of the unique mapping that produces the time series.

When the autonomous system generates samples, past values are used as inputs to generate the following samples, which means that the training should constrain also the iterates of the predictive mapping. Putting it in a simple way, we should train the predictor in the same way we are going to use it for testing (i.e. as an autonomous system).

We propose multistep prediction (or trajectory learning) as the way to constrain the iterates of the mapping developed by the predictor. Let us define

$$E = \sum_{i=2m+1}^{k} dist(x(i+1) - \tilde{x}(i+1))$$ (4)

where k is the number of prediction steps (length of the trajectory) and $\tilde{x}(i+1)$ is an estimate of the predictive map

$$\tilde{x}(i+1) = \tilde{F}^{\perp}(\hat{x}(i-2m),...,\hat{x}(i))$$ (5)

with

$$\hat{x}(i) = \left[ \begin{array}{ll} x(i) & 0 \le i \le 2m \\ \tilde{F}^{\perp}(x(i-2m-1),...,x(i-1)) & i > 2m \end{array} \right.$$

Equation (5) states that $\tilde{x}(i)$ is the i-2m iterate of the predictive part of the map (for i>2m), i.e.

$$\tilde{x}(i+1) = (\tilde{F}^{\perp}(\tilde{F}^{\perp}(...\tilde{F}^{\perp}(\hat{x}(2m))))) = (\tilde{F}^{\perp}(\hat{x}(2m)))^{i-2m} \qquad (6)$$

Hence, minimizing the criterion expressed by equation (4) an optimal multistep predictor is obtained. The number of constraints that are imposed during learning is associated with k, the number of prediction steps, which corresponds to the number of iterations of the map. The more iterations, the less likely a sub-optimal solution is found, but note that the training time is being proportionally increased. In a chaotic time series there is a more important consideration that must be brought into the picture, the divergence of nearby trajectories, as we are going to see in a following section.

## 3. Multistep prediction with neural networks

Figure 1 shows the topology proposed in [4] to identify the nonlinear mapping. Notice that the proposed topology is a *recurrent neural network, with a global feedback loop*. This topology was selected to allow the training of the predictor in the same way as it will be used in testing, i.e. using the previous network outputs to predict the next point. This recurrent architecture should be trained with a mechanism that will constrain the iterates of the map as was discussed above. Single step prediction does not fit this requirement.

With multistep prediction, the model system can be trained in the same way as it is used in testing. We seed the dynamic net with a set of input samples, disconnect the input and feed back the predicted sample to the input for k steps. The mean square error between the predicted and true sample at each step is used as the cost function (equation (4)). If the network topology was feedforward, batch learning could be used to train the network, and static backpropagation applied to train the net. However, as a recurrent topology is utilized, a learning paradigm such as backpropagation through time (BPTT) or real time recurrent learning (RTRL) must be utilized [6]. The use of these training methods should not come as a surprise since we are in fact fitting a trajectory over time, so the gradients are time varying. This learning method is sometimes called "trajectory learning" in the recurrent learning literature [6]. A criterion to select the length of the trajectory k will be presented below.

The procedure described above must be repeated for several different segments of the time series. For each new training segment, 2m+1 samples of the original time series are used to seed the predictor. To ease the training we suggest that successive training sequences of length k overlap by q samples (q<k). For chaotic time series we also suggest that the error be weighted according to the largest Lyapunov exponent. Hence

the cost function becomes

$$E = \sum_{j=0}^{r} \sum_{i=2m+1}^{k} h(i) \cdot dist(x(i+jq+1) - \tilde{x}(i+jq+1)) \qquad (7)$$

where r is the number of training sequences, and

$$h(i) = (e^{\lambda_{max}\Delta t})^{-(i-2m-1)} \qquad (8)$$

In this equation $\lambda_{max}$ is the largest Lyapunov exponent and $\Delta t$ the sampling interval. With this weighting the errors for later iteration are given less credit, as they should since due to the divergence of trajectories a small error is magnified proportionally to the largest Lyapunov exponent [7].

# 4. Finding the length of the trajectory

From the point of view of dynamic modelling, each training sequences should preferably contain enough information to model the attractor. This means that each sequence should be no shorter than the orbital length around the attractor. We proposed to estimate the orbital length as the reciprocal of the median frequency of the spectrum of the time series [8]. Basically this quantity is the average time required for a point to return to the same neighborhood in the attractor.

The length of the trajectory is also equivalent to the number of constraints we impose on the iterative map describing the dynamical model. However, in a chaotic time series there is another fundamental limitation imposed on the trajectory length - the natural divergence of trajectories which is controlled by $\lambda_{max}$, the largest Lyapunov exponent. If the trajectory length is too long, then instabilities in the training can be expected. A full discussion of this topic is beyond the scope of this paper, and is presented elsewhere [8]. We just want to say that when $\lambda_{max}$ is positive there is an uncertainty region around each predicted point that is a function of the number of prediction steps (due to cummulative error). If the trajectory length is too long the uncertainty regions from two neighboring trajectories will overlap, creating conflicting requirements for training (the model is requried to develop a map to follow both segments A and B- Figure 2).

It turns out that one can approximately find the number of iterations $i_s$ that will guarantee no overlap of uncertainty regions [8]. The length of the principal axis of the uncertainty region around a signal trajectory at iteration i can be estimated as

$$\varepsilon_i = \varepsilon_0 e^{\lambda_{max}i\Delta t} \qquad (9)$$

where $\varepsilon_0$ is the initial separation. Now assuming that the two principal axis of nearby trajectories are colinear, we should choose the number of iterations $i_s$ such that the distance $d_i$ between trajectories is larger than the uncertainty region, i.e. $d_{i_s} \geq 2\varepsilon_{i_s}$.

The estimate of $i_s$ should be averaged over a number of neighboring training sequences (~50 depending on the signal dynamics).

Hence, to apply this method three quantities must be estimated: the largest Lyapunov

exponent, using one of the accepted algorithms. The initial separation can be estimated from the one-step predictor. And $i_s$ by averaging local divergence. The computation time required to estimate these quantities is usually much less than setting by trial and error the length of the trajectory until a reasonable learning curve is achieved.

We also developed a method to train predictors for chaotic signals with large $\lambda_{max}$, but it will not be covered in this paper [8].

# 5. Results

We used this methodology to model the Mackey-Glass system (d=30, sampled at 1/6 Hz). A signal of 500 samples was obtained by 4th order Runge-Kutta integration and normalized between -1,1. The largest Lyapunov exponent for this signal is 0.0071 nats/sec. We selected a time delay neural network (TDNN) with topology 8-14-1. The output unit is linear, and the hidden layer has sigmoid nonlinearities. The number of taps in the delay line is 8.

We trained a one-step predictor and the multistep predictor with the methodology developed in this paper to compare results. The single step predictor was trained with static backpropagation with no momentum and step size of 0.001. Trained was stopped after 500 iterations. The final MSE was 0.000288. After training, the predictor was seeded with the first 8 points of the time series and iterated for 3,000 times. Figure 3a shows the corresponding output. Notice that the waveform produced by the model is much more regular that the Mackey-Glass signal, showing that some fine detail of the attractor has not been captured.

Next we trained the same TDNN with a global feedback loop (TDNNGF). The estimate of the $i_s$ over the neighboring orbits provided an estimate of 14, and it is taken as the length of the trajectory. We displaced each training sequence by 3 samples (q=3 in eqn 7). BPTT was used to train the TDNNGF for 500 iterations over the same signal. The final MSE was 0.000648, higher than for the TDNN case. We could think that the resulting predictor was worse. The TDNNGF predictor was initialized with the same 8 samples of the time series and iterated for 3,000 times. Figure 3b shows the resulting waveform. It "looks" much closer to the original Mackey-Glass time series. We computed the average prediction error as a function of iteration for both predictors and also the theoretical rate of divergence of trajectories assuming an initial error $\varepsilon_0$ (Casdagli conjecture, which is the square of eqn 9) [7]. As can be seen in Figure 4 the TDNNGF is much closer to the theoretical limit, which means a much better model. We also computed the correlation dimension and the Lyapunov exponent estimated from the generated time series, and the figures obtained from TDNNGF are closer to the original time series.

Figure 5 shows the instability present in the training when the trajectory length is above the estimated value of 14. For this case the trajectory length is 20. As can be seen the MSE decreases but then fluctuates showing instability in the training.

# 6. Conclusions

This paper addresses dynamic modelling with artificial neural networks. We showed

that the network topology should be recurrent such that the iterative map is constrained during learning. This is a necessity since dynamic modelling seeks to capture the long term behavior of the dynamical system. These models can also be used as a sample by sample predictors. Since the network topology is recurrent, backpropagation through time or real time recurrent learning should be used in training. In this paper we showed how to select the length of the trajectory to avoid instability during training.

A lot more work needs to be done to reliably capture dynamical properties of time series and encapsulate them in artificial models. But we believe that the careful analysis of the dynamic characteristics and the study of its impact on the predictive model performance is much more promising than guess work. According to this (and others) studies, modelling of chaotic time series of low $\lambda_{max}$ seems a reality. We have extended some of this work for time series with larger $\lambda_{max}$, and successfully captured the dynamics of the Lorenz system [8]. But there, the parameters for learning have to be much more carefully selected, and some of the choices are still arbitrary. The main issue is that the trajectories diverge so rapidly that predictors have a hard time to capture information regarding the global system dynamics. It is interesting to study the limit of predictability of this type of approach for high dimensional and high $\lambda_{max}$ chaos.

| Predictor | Corr. Dim. | Lyapunov |
|---|---|---|
| MG30 | 2.70+/-0.05 | 0.0073+/-0.0001 |
| TDNNGF | 2.65+/-0.03 | 0.0074+/-0.0001 |
| TDNN | 1.60+/-0.10 | 0.0063+/-0.0001 |

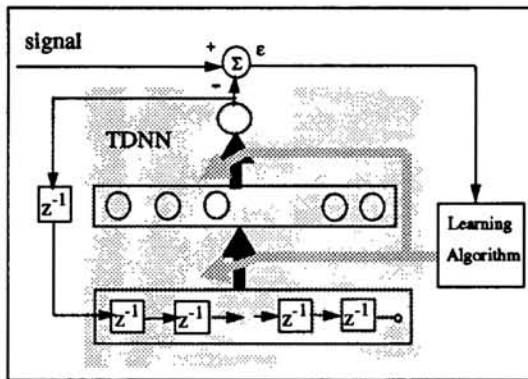

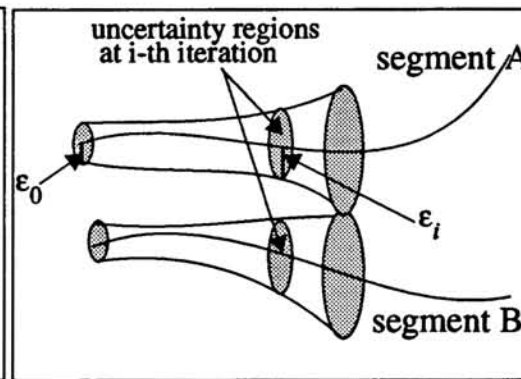

Figure 1. Proposed recurrent architecture (TDNNGF)    Figure 2. State space representation in training a model

# 7. Acknowledgments

This work was partially supported by NSF grant #ECS-9208789, and ONR #1494-94-1-0858.

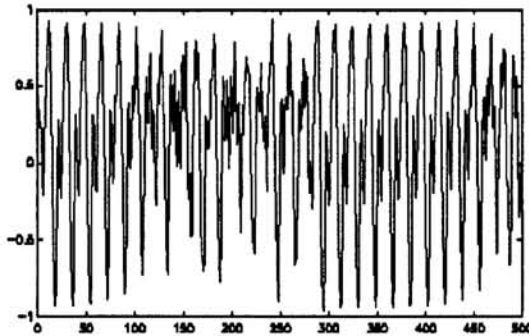

Figure 3a. Generated sequence with the TDNN

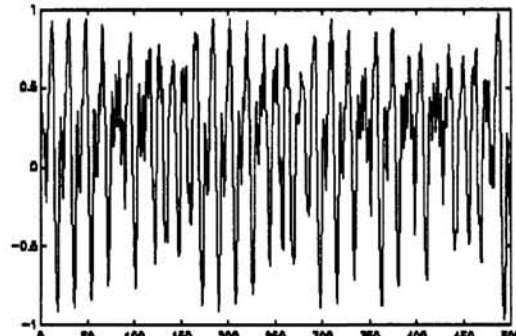

Figure 3b. Generated sequence with the TDNNGF

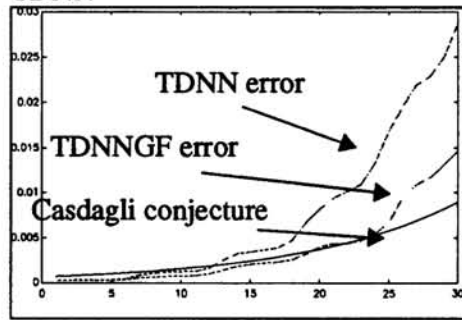

Figure 4. Comparison of predictors

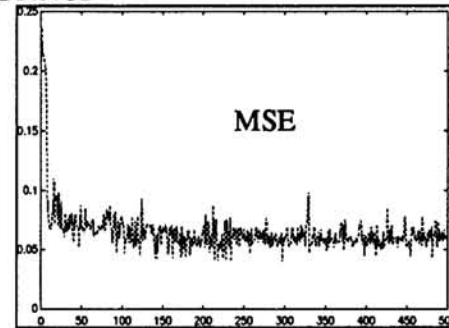

Figure 5. Instability in training

## 8. References

[1] Box, G. E., and G. M. Jenkins, Time Series Analysis, Forecasting and Control, Holden Day, San Francisco, 1970.

[2] Weigend, A. S., B. A. Huberman, and D. E. Rumelhart, "Predicting the future: a connectionist approach," International Journal of Neural Systems, vol. 1, pp. 193-209, 1990.

[3] Lapedes, R., and R. Farber, "Nonlinear signal processing using neural networks: prediction and system modelling," Technical Report LA-UR87-2662, Los Alamos National Laboratory, Los Alamos, New Mexico, 1987.

[4] Kuo J-M., Principe J.C., "A systematic approach to chaotic time series modeling with neural networks", in IEEE Workshop on Neural Nets for Signal Processing, Ermioni, Greece, 1994.

[5] Principe, J. C., A. Rathie, and J. M. Kuo, "Prediction of chaotic time series with neural networks and the issue of dynamic modeling," International Journal of Biburcation and Chaos, vol. 2, no. 4, pp. 989-996, 1992.

[6] Hertz, J, A. Krogh, and R. G. Palmer, Introduction to the Theory of Neural Computation, Addison-Wesley, Redwood City, CA, 1991.

[7] Casdagli, M., "Nonlinear prediction of chaotic time series," Physica D 35, pp.335-356, 1989.

[8] Kuo, J.M., "Nonlinear Dynamic Modelling with Artificial neural networks", Ph.D. dissertation, University of FLorida, 1993.
